# Leveraged Vector Machines

**Yoram Singer**
Hebrew University
singer@cs.huji.ac.il

## Abstract

We describe an iterative algorithm for building vector machines used in classification tasks. The algorithm builds on ideas from support vector machines, boosting, and generalized additive models. The algorithm can be used with various continuously differential functions that bound the discrete (0-1) classification loss and is very simple to implement. We test the proposed algorithm with two different loss functions on synthetic and natural data. We also describe a norm-penalized version of the algorithm for the exponential loss function used in AdaBoost. The performance of the algorithm on natural data is comparable to support vector machines while typically its running time is shorter than of SVM.

## 1 Introduction

Support vector machines (SVM) [1, 13] and boosting [10, 3, 4, 11] are highly popular and effective methods for constructing linear classifiers. The theoretical basis for SVMs stems from Vapnik's seminal on learning and generalization [12] and has proved to be of great practical usage. The first boosting algorithms [10, 3], on the other hand, were developed to answer certain fundamental questions about PAC-learnability [6]. While mathematically beautiful, these algorithms were rather impractical. Later, Freund and Schapire [4] developed the AdaBoost algorithm, which proved to be a practically useful meta-learning algorithm. AdaBoost works by making repeated calls to a *weak learner*. On each call the weak learner generates a single *weak hypothesis*, and these weak hypotheses are combined into an ensemble called *strong hypothesis*. Recently, Schapire and Singer [11] studied a simple generalization of AdaBoost in which a weak-hypothesis can assign a real-valued *confidence* to each prediction. Even more recently, Friedman, Hastie, and Tibshirani [5] presented an alternative view of boosting from a statistical point of view and also described a new family of algorithms for constructing generalized additive models of base learners in a similar fashion to AdaBoost. The work of Friedman, Hastie, and Tibshirani generated lots of attention and motivated research in classification algorithms that employ various loss functions [8, 7].

In this work we combine ideas from the research mentioned above and devise an alternative approach to construct vector machines for classification. As in SVM, the base predictors that we use are Mercer kernels. The value of a kernel evaluated at an input pattern, i.e., the dot-product between two instances embedded in a high-dimensional space, is viewed as a real-valued prediction. We describe a simple extension to additive models in which the prediction of a base-learner is a linear transformation of a given kernel. We then describe an iterative algorithm that greedily adds kernels. We derive our algorithm using the exponential loss function used in AdaBoost and the loss function used by Friedman, Hastie, and Tibshirani [5] in "LogitBoost". For brevity we call the resulting classifiers boosted vector machines (BVM) and logistic vector machines (LVM). We would like to note in passing

that the resulting algorithms are *not* boosting algorithms in the PAC sense. For instance, the weak-learnability assumption that the weak-learner can always find a weak-hypothesis is violated. We therefore adopt the terminology used in [2] and call the resulting classifiers *leveraged vector machines*.

The leveraging procedure we give adopts the chunking technique from SVM. After presenting the basic leveraging algorithms we compare their performance with SVM on synthetic data. The experimental results show that the leveraged vector machines achieve similar performance to SVM and often the resulting vector machines are smaller than the ones obtained by SVM. The experiments also demonstrate that BVM is especially sensitive to (malicious) label noise while LVM seems to be more insensitve. We also describe a simple norm-penalized extension of BVM that provides a partial solution to overfitting in the presence of noise. Finally, we give results of experiments performed with natural data from the UCI repository and conclude.

## 2 Preliminaries

Let $S = \langle (x_1, y_1), \ldots, (x_m, y_m) \rangle$ be a sequence of training examples where each *instance* $x_i$ belongs to a *domain* or *instance space* $\mathcal{X}$, and each *label* $y_i$ is in $\{-1, +1\}$. (The methods described in this paper to build vector machines and SVMs can be extended to solve multiclass problems using, for instance, error correcting output coding. Such methods are beyond the scope of this paper and will be discussed elsewhere). For convenience, we will use $\tilde{y}_i$ to denote $(y_i + 1)/2 \in \{0, 1\}$.

As is boosting, we assume access to a *weak* or *base* learning algorithm which accepts as input a *weighted* sequence of training examples $S$. Given such input, the weak learner computes a weak (or base) *hypothesis* $h$. In general, $h$ has the form $h : \mathcal{X} \rightarrow \mathbb{R}$. We interpret the sign of $h(x)$ as the predicted label ($-1$ or $+1$) to be assigned to instance $x$, and the magnitude $|h(x)|$ as the "confidence" in this prediction.

To build vector machines we use the notion of confidence-rated predictions, take for base hypotheses sample-based Mercer kernels [13], and define the confidence (i.e., the magnitude of prediction) of a base learner to be the value of its dot-product with another instance. The sign of the prediction is set to be the label of the corresponding instance. Formally, for each base hypothesis $h$ there exist $(x_j, y_j) \in S$ such that $h(x) = y_j K(x_j, x)$ and $K(u, v)$ defines an inner product in a feature space: $K(u, v) = \sum_{k=1}^{\infty} a_k \psi_k(u) \psi_k(v)$. We denote the function induced by an instance label pair $(x_j, y_j)$ with a kernel $K$ by $\phi_j(x) = y_j K(x_j, x)$. Our goal is to find a classifier $f(x)$, called a strong hypothesis in the context of boosting algorithms, of the form $f(x) = \sum_{t=1}^{T} \alpha_t h_t(x) + \beta$, such that the signs of the predictions of the classifier should agree, as much as possible, with the labels of the training instances.

The leverage algorithm we describe maintains a distribution $D$ over $\{1, \ldots, m\}$, i.e., over the indices of $S$. This distribution is simply a vector of non-negative weights, one weight per example and is an exponential function of the classifier $f$ which is built incrementally,

$$D(i) = \frac{1}{Z} \exp\left(-y_i f(x_i)\right) \text{ where } Z = \sum_{i=1}^{m} \exp\left(-y_i f(x_i)\right). \tag{1}$$

For a random function $g$ of the input instances and the labels, we denote the *sample* expectation of $g$ according to $D$ by $E_D(g) = \sum_{i=1}^{m} D(i) g(x_i, y_i)$. We also use this notation to denote the expectation of matrices of random functions. We will convert a confidence-rated classifier $f$ into a randomized predictor by using the soft-max function and denote it by $P(x_i)$ where

$$P(x_i) = \frac{\exp\left(f(x_i)\right)}{\exp\left(f(x_i)\right) + \exp\left(-f(x_i)\right)} = \frac{1}{1 + \exp\left(-2f(x_i)\right)}. \tag{2}$$

## 3   The leveraging algorithm

The basic procedure to construct leveraged vector machines builds on ideas from [11, 5] by extending the prediction to be a linear function of the base classifiers. The algorithm works in rounds, constructing a new classifier $f_t$ from the previous one $f_{t-1}$ by adding a new base hypothesis $h_t$ to the current classifier, $f_t$. Denoting by $D_t$ and $P_{t+1}$ the distribution and probability given by Eqn. (1) and Eqn. (2) using $f_t$ and $f_{t+1}$, the algorithm attempts to minimize either the exponential function that arise in AdaBoost:

$$
\begin{aligned}
Z &= \sum_{i=1}^{m} \exp\left(-y_i f_t(x_i)\right) = \sum_{i=1}^{m} \exp\left(-y_i(f_{t-1}(x_i) + \alpha_t h_t(x_i) + \beta_t)\right) \\
&\sim \sum_{i=1}^{m} D_t(i) \exp\left(-y_i(\alpha_t h_t(x_i) + \beta_t)\right),
\end{aligned} \tag{3}
$$

or the logistic loss function:

$$
\begin{aligned}
L &= \sum_{i=1}^{m} \log\left(1 + \exp\left(-2y_i f_t(x_i)\right)\right) \tag{4} \\
&= \sum_{i=1}^{m} \log\left(1 + \exp\left(-2y_i(f_{t-1}(x_i) + \alpha_t h_t(x_i) + \beta_t)\right)\right) \\
&= -\sum_{i=1}^{m} \left(\tilde{y}_i \log(P_{t+1}(x_i)) + (1 - \tilde{y}_i)\log(1 - P_{t+1}(x_i))\right). \tag{5}
\end{aligned}
$$

We initialize $f_0(x)$ to be zero everywhere and run the procedure for a predefined number of rounds $T$. The final classifier is therefore $f_T(x) = \sum_{t=1}^{T}(\alpha_t h_t(x) + \beta_t) = \beta + \sum_{t=1}^{T} \alpha_t h_t(x)$  where  $\beta = \sum_t \beta_t$ . We would like to note parenthetically that it is possible to use other loss functions that bound the 0-1 (classification) loss (see for instance [8]). Here we focus on the above loss functions, $L$ and $Z$. Fixing $f_{t-1}$ and $h_t$, these functions are convex in $\alpha_t$ and $\beta_t$ which guarantees, under mild conditions (details omitted due to lack of space), the uniqueness of $\alpha_t$ and $\beta_t$.

On each round we look for the current base hypothesis $h_t$ that will reduce the loss function ($Z$ or $L$) the most. As discussed before, each input instance $x_j$ defines a function $\phi_j(x)$ and is a candidate for $h_t(x)$. In general, there is no close form solution for Eqn. (3) and (5) and finding $\alpha$ and $\beta$ for each possible input instance is time consuming. We therefore use a quadratic approximation for the loss functions. Using the quadratic approximation, for each $\phi_j$ we can find $\alpha$ and $\beta$ analytically and calculate the reduction in the loss function. Let $\nabla Z = (\frac{\partial Z}{\partial \alpha}, \frac{\partial Z}{\partial \beta})^T$ and $\nabla L = (\frac{\partial L}{\partial \alpha}, \frac{\partial L}{\partial \beta})^T$ be the column vectors of the partial derivatives of $Z$ and $L$ w.r.t $\alpha$ and $\beta$ (fixing $f_{t-1}$ and $h_t$). Similarly, let $\nabla^2 Z$ and $\nabla^2 L$ be the $2 \times 2$ matrices of second order derivatives of $Z$ and $L$ with respect to $\alpha$ and $\beta$. Then, quadratic approximation yields that $(\alpha, \beta)^T = (\nabla^2 Z)^{-1} \nabla Z$ and $(\alpha, \beta)^T = (\nabla^2 L)^{-1} \nabla L$. On each round $t$ we maintain a distribution $D_t$ which is defined from $f_t$ as given by Eqn. (1) and conditional class probability estimates $P_t(x_i)$ as given by Eqn. (2). Solving the linear equation above for $\alpha$ and $\beta$ for each possible instance is done by setting $h_t(x) = \phi_j(x)$, we get for $Z$

$$
\left( \begin{array}{c} \alpha_j \\ \beta_j \end{array} \right) = \left[ E_{D_t} \left( \begin{array}{cc} \phi_j^2 & \phi_j \\ \phi_j & 1 \end{array} \right) \right]^{-1} E_{D_t} \left[ y \left( \begin{array}{c} \phi_j \\ 1 \end{array} \right) \right], \tag{6}
$$

and for $L$

$$
\left( \begin{array}{c} \alpha_j \\ \beta_j \end{array} \right) = \frac{1}{2} \left[ E_{D_t} \left[ P(1-P) \left( \begin{array}{cc} \phi_j^2 & \phi_j \\ \phi_j & 1 \end{array} \right) \right] \right]^{-1} E_{D_t} \left[ (\tilde{y} - P) \left( \begin{array}{c} \phi_j \\ 1 \end{array} \right) \right]. \tag{7}
$$

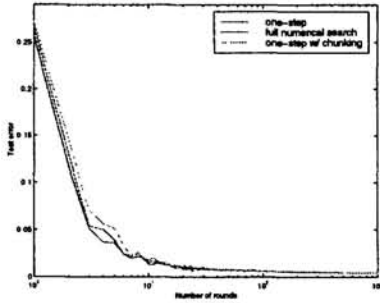

Figure 1: Comparison of the test error as a function of number of leveraging rounds when using full numerical search for $\alpha$ and $\beta$, a "one-step" numerical search based on a quadratic approximation of the loss function, and a one-step search with chunking of the instances.

Note that the equations above share much in common and require, after pre-computing $P(x_i)$, the same amount of computation time.

After calculating the value of $\alpha$ and $\beta$ for each instance $(x_j, y_j)$, we simply evaluate the corresponding value of the loss function, choose the instance $(x_{j^*}, y_{j^*})$ that attains the minimal loss, and set $h_t = \phi_{j^*}$. We then numerically search for the optimal value of $\alpha$ and $\beta$ by iterating Eqn. (6) or Eqn. (7) and summing the values into $\alpha_t$ and $\beta_t$. We would like to note that typically two or three iterations suffice and we can save time by using the value of $\alpha$ and $\beta$ found using the quadratic approximation without a full numerical search for the optimal value of $\alpha$ and $\beta$. (See also Fig. 1.) We repeat this process for $T$ rounds or until no instance can serve as a base hypothesis. We note that the same instance can be chosen more than once, although not in consecutive iterations, and typically only a small fraction of the instances is actually used in building $f$. Roughly speaking, these instances are the "support patterns" of the leveraged machines although they are not necessarily the geometric support patterns.

As in SVMs, in order to make the search for a base hypothesis efficient we pre-compute and store $K(x, x')$ for all pairs $x \neq x'$ from $S$. Storing these values require $|S|^2$ space, which might be prohibited in large problems. To save space, we employ the idea of chunking used in SVM. We partition $S$ into $r$ blocks $S_1, S_2, \ldots, S_r$ of about the same size. We divide the iterations into sub-groups such that all iterations belonging to the $i$th sub-group use and evaluate kernels based on instances from the $i$th block only. When switching to a new block $k$ we need to compute the values $K(x, x')$ for $x \in S$ and $x' \in S_k$. This division into blocks might be more expensive since we typically use each block of instances more than once. However, the storage of the kernel values can be done in place and we thus save a factor of $r$ in memory requirements. In practice we found that chunking does not hurt the performance. In Fig. 1 we show the test error as a function of number of rounds when using (a) full numerical search to determine $\alpha$ and $\beta$ on each round, (b) using the quadratic approximation ("one-step") to find $\alpha$ and $\beta$, and (c) using quadratic approximation with chunking. The number of instances in the experiment is 1000, each block for chunking is of size 100, and we switch to a different block every 100 iterations. (Further description of the data is given in the next section.) In this example, after 10 iterations, there is virtually no difference in the performance of the different schemes.

## 4 Experiments with synthetic data

In this section we describe experiments with synthetic data comparing different aspects of leveraged vector machines to SVMs. The original instance space is two dimensional where the positive class includes all points inside a circle of radius $R$, i.e., an instance $(u_1, u_2) \in \mathbb{R}^2$ is labeled $+1$ iff $u_1^2 + u_2^2 \leq R$. The instances were picked at random according to a zero mean unit variance normal distribution and $R$ was set such exactly half of the instances belong to the positive class. In all the experiments described in this section we generated 10 groups of training and test sets each of which includes 1000 train and test examples. Overall, there are $10,000$ training examples and $10,000$ test examples. The

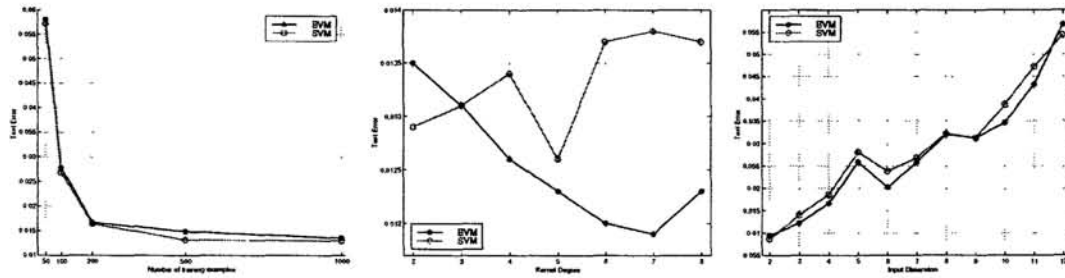

Figure 2: Performance comparison of SVM and BVM as a function of the training data size (left), the dimension of the kernels (middle), and the number of redundant features.

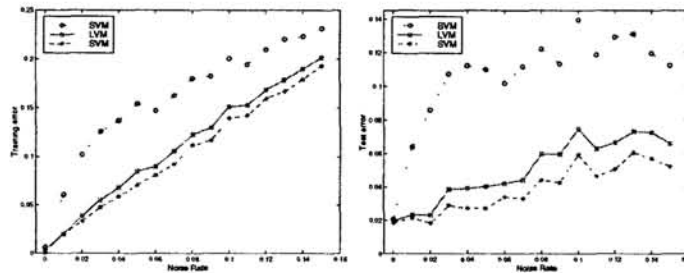

Figure 3: Train and test errors for SVM, LVM, and BVM as a function of the label noise.

average variance of the estimates of the empirical errors across experiments is about 0.2%. For SVM we set the regularization parameter, $C$, to 100 and used 500 iterations to build leveraged machines. In all the experiments without noise the results for BVM and LVM were practically the same. We therefore only compare BVM to SVM in Fig. 2. Unless said otherwise we used polynomials of degree two as kernels: $K(x,'x) = (x \cdot x' + 1)^2$. Hence, the data is separable in the absence of noise.

In the first experiment we tested the sensitivity to the number of training examples by omitting examples from the training data (without any modification to the test sets). On the left part of Fig. 2 we plot the test error as a function of the number of training examples. The test error of BVM is almost indistinguishable from the error of SVM and performance of both methods improves very fast as a function of training examples. Next, we compared the performance as a function of the dimension of polynomial constituting the kernel. We ran the algorithms with kernels of the form $K(x,'x) = (x \cdot x' + 1)^d$ for $d = 2, \ldots, 8$. The results are depicted in the middle plots of Fig. 2. Again, the performance of BVM and SVM is very close (note the small scale of the $y$ axis for the test error in this experiment). To conclude the experiments with clean, realizable, data we checked the sensitivity to irrelevant features of the input. Each input instance $(u_1, u_2)$ was augmented with random elements $u_3, \ldots, u_l$ to form an input vector of dimension $l$. The right hand side graphs of Fig. 2 shows the test error as a function of $l$ for $l = 2, \ldots, 12$. Once more we see that the performance of both algorithms is very similar.

We next compared the performance of the algorithms in the presence of noise. We used kernels of dimension two and instances without redundant features. The label of each instance was flipped with probability $\epsilon$. We ran 15 sets of experiments, for $\epsilon = 0.01, \ldots, 0.15$. As before, each set included 10 runs each of which used 1000 training examples and 1000 test examples. In Fig. 3 we show the average training error (left), and the average test error (right), for each of the algorithms. It is apparent from the graphs that BVMs built based on the exponential loss are much more sensitive to noise than SVMs and LVMs, and their generalization error degrades significantly, even for low noise rates. The generalization error of LVMs is, on the other hand, only slightly worse than the that of SVMs, although the

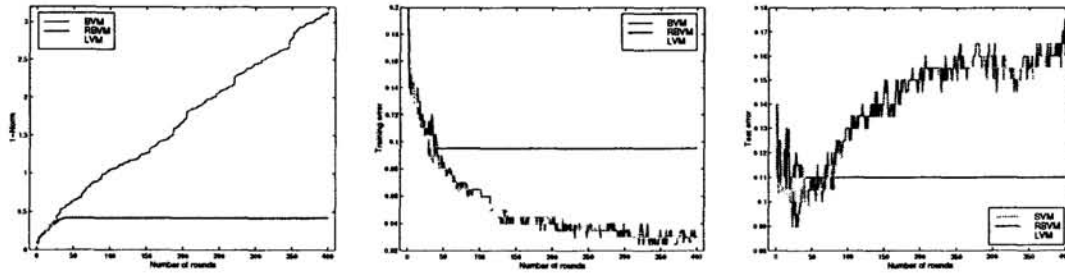

Figure 4: The training error, test error, and the cumulative $L_1$ norm ($\sum_{t'=1}^{t} |\alpha_t'|$) as a function of the number of leveraging iterations for LVM,BVM, and PBVM.

only algorithmic difference in constructing BVMs and LVMs is in the loss function. The fact that LVMs exhibit performance similar to SVM can be partially attributed to the fact that the asymptotic behavior of their loss functions is the same.

## 5   A norm-penalized version

One of the problems with boosting and the corresponding leveraging algorithm with the exponential loss described here, is that it might increase the confidence on a few instances while misclassifying many other instances, albeit with a small confidence. This often happens on late rounds, during which the distribution $D_t(i)$ is concentrated on a few examples, and the leveraging algorithm typically assigns a large weight to a weak hypothesis that does not effect most of the instances. It is therefore desired to control the complexity of the leveraged classifiers by limiting the magnitude of base hypotheses' weights. Several methods have been proposed to limit the confidence of AdaBoost, using, for instance, regularization (e.g., [9]) or "smoothing" the predictions [11]. Here we propose a norm-penalized method for BVM that is very simple to implement and maintains the convexity properties of the objective function. Following the idea Cortes and Vapnik's of SVMs in the non-separable case [1] we add the following penalization term: $\gamma_0 \exp\left(\sum_{t=1}^{T} |\alpha_t|^p\right)$ . Simple algebric manipulation implies that the objective function at the $t$th round for BVMs with the penalization term above is,

$$\tilde{Z}_t = \sum_{i=1}^{m} D_t(i) \exp\left(-y_i(\alpha_t h_t(x_i) + \beta_t)\right) + \gamma_t \exp(|\alpha_t|^p) . \tag{8}$$

It is also easy to show that the penalty parameter should be updated after each round is: $\gamma_t = \gamma_{t-1} \exp(|\alpha_{t-1}|^p)/Z_{t-1}$. Since $Z_t < 1$, unless there is no kernel function better than random, $\gamma_t$ typically increases as a function of $t$, forcing more and more the new weights to be small. Note that Eqn. (8) implies that the search for a base predictor $h_t$ and weights $\alpha_t, \beta_t$ on each round can still be done independently of previous rounds by maintaining the distribution $D_t$ and a single regularization value $\gamma_t$. The penalty term for $p = 1$ and $p = 2$ simply adds a diagonal term to the matrix of second order derivatives (Eqn. (6)) and the algorithm follows the same line (details omitted). For brevity we call the norm-penalized leveraging procedure PBVM. In Fig. 4 we plot the test error (right), training error (middle), and $\sum_t |\alpha_t|$ as functions of number of rounds for LVM, BVM, and PBVM with $p = 1$ $\gamma_0 = 0.01$. The training set in this example was made small on purpose (200 examples) and was contaminated with 5% label noise. In this very small example both LVM and BVM overfit while PBVM stops increasing the weights and finds a reasonably good classifier. The plots demonstrate that the norm-penalized version can safeguard against overfitting by preventing the weights from growing arbitrarily large, and that the effect of the penalized version is very similar to early stopping. We would like

| Data Set (Source) | #Example & #Feature | SVM Size | LVM Size | BVM Size | RBVM Size | SVM Error | LVM Error | BVM Error | PBVM Error |
|---|---|---|---|---|---|---|---|---|---|
| labor (uci) | 57 : 16 | 12.5 | 13.7 | 16.1 | 13.6 | **6.0** | 14.0 | 14.0 | 12.0 |
| echocard. (uci) | 74 : 12 | 7.8 | 13.0 | 12.6 | 12.4 | 8.6 | **5.7** | 10.0 | 10.0 |
| bridges (uci) | 102 : 7 | 27.2 | 20.2 | 18.5 | 17.9 | 15.0 | 15.0 | 23.0 | **14.0** |
| hepatitis (uci) | 155 : 19 | 41.2 | 13.5 | 17.4 | 14.0 | **21.3** | 22.0 | 22.7 | 22.0 |
| horse-colic (uci) | 300 : 23 | 122.0 | 13.0 | 13.0 | 13.0 | 14.7 | 14.7 | 14.7 | **13.2** |
| liver (uci) | 345 : 6 | 228.6 | 11.3 | 12.8 | 10.7 | 33.8 | 35.6 | **33.5** | 35.6 |
| ionosphere (uci) | 351 : 34 | 63.4 | 58.9 | 67.9 | 59.1 | 13.7 | **13.1** | 16.9 | 13.7 |
| vote (uci) | 435 : 16 | 37.0 | 37.0 | 41.0 | 37.0 | **4.4** | 5.2 | 5.9 | 5.2 |
| ticket1 (att) | 556 : 78 | 48.1 | 84.6 | 89.3 | 82.3 | 8.4 | **3.3** | 11.5 | 5.1 |
| ticket2 (att) | 556 : 53 | 52.6 | 77.1 | 75.4 | 74.0 | 6.6 | **6.4** | 8.0 | **6.4** |
| ticket3 (att) | 556 : 61 | 46.1 | 76.2 | 77.8 | 73.3 | 6.9 | **4.9** | 7.6 | 6.7 |
| bands (uci) | 690 : 39 | 265.5 | 78.2 | 76.4 | 75.6 | **32.8** | 33.2 | 34.3 | 33.3 |
| breast-wisc (uci) | 699 : 9 | 49.3 | 26.5 | 24.4 | 24.0 | **3.5** | 3.6 | 4.1 | 4.1 |
| pima (uci) | 768 : 8 | 360.7 | 47.7 | 30.3 | 22.8 | 23.0 | 22.6 | 23.2 | **22.1** |
| german (uci) | 1000 : 10 | 485.2 | 89.8 | 96.5 | 87.0 | **23.5** | 24.0 | 23.8 | 24.1 |
| weather (uci) | 1000 : 35 | 562.0 | 52.0 | 52.0 | 52.0 | 25.9 | **25.4** | **25.4** | **25.4** |
| network (att) | 2600 : 35 | 1031.0 | 42.0 | 43.0 | 42.0 | 24.8 | **21.2** | 23.5 | **21.2** |
| splice (uci) | 3190 : 60 | 318.0 | 153.0 | 156.0 | 153.0 | **8.0** | 8.4 | 8.4 | 8.4 |
| boa (att) | 5000 : 68 | 637.0 | 183.0 | 178.0 | 160.0 | 41.5 | **40.8** | **40.8** | 41.0 |

Table 1: Summary of results for a collection of binary classification problems.

to note that we found experimentally that the norm-penalized version does compensate for incorrect estimates of $\alpha$ and $\beta$ due to malicious label noise. The experimental results given in the next section show, however, that it does indeed help in preventing overfitting when the training set is small.

## 6 Experiments with natural data

We compared the practical performance of leveraged vector machines with SVMs on a collection of nineteen dataset from the UCI machine learning repository and AT&T networking and marketing data. For SVM we set $C = 100$. We built each of the leveraged vector machines using 500 rounds. For PBVM we used again $p = 1$ and $\gamma_0 = 0.01$. We used chunking in building the leveraged vector machines, dividing each training set into 10 blocks. For all the datasets, with the exception of "boa", we used 10-fold cross validation to calculate the test error. (The dataset "boa" has 5000 training examples and 6000 test examples.) The performance of SVM, LVM, and PBVM seem comparable. In fact, with the exception of a very few datasets the differences in error rates are *not* statistically significant. Of the three methods (SVM, PBVM, and LVM), LVM is the simplest to implement the time required to build an LVM is typically much shorter than that of an SVM. It is also worth noting that the size of leveraged machines is often smaller than the size of the corresponding SVM. Finally, it apparent that PBVMs frequently yield better results than BVMs, especially for small and medium size datasets.

## References

[1] Corinna Cortes and Vladimir Vapnik. Support-vector networks. *Machine Learning*, 20(3):273–297, September 1995.

[2] N. Duffy and D. Helmbold. A geometric approach to leveraging weak learners. EuroCOLT '99.

[3] Yoav Freund. Boosting a weak learning algorithm by majority. *Information and Computation*, 121(2):256–285, 1995.

[4] Yoav Freund and Robert E. Schapire. A decision-theoretic generalization of on-line learning and an application to boosting. *Journal of Computer and System Sciences*, 55(1):119–139, August 1997.

[5] J. Friedman, T. Hastie, and R. Tibshirani. Additive logistic regression: a statistical view of boosting. Tech. Report, 1998.

[6] Michael Kearns and Leslie G. Valiant. Cryptographic limitations on learning Boolean formulae and finite automata. *Journal of the Association for Computing Machinery*, 41(1):67–95, January 1994.

[7] John D. Lafferty. Additive models, boosting and inference for generalized divergences. In *Proceedings of the Twelfth Annual Conference on Computational Learning Theory*, 1999.

[8] L. Mason, J. Baxter, P. Bartlett, and M. Frean. Doom II. Technical report, Depa. of Sys. Eng. ANU 1999.

[9] G. Rätsch, T.Onoda, and K.-R. Müller. Regularizing adaboost. In *Advances in Neural Info. Processing Systems 12*, 1998.

[10] Robert E. Schapire. The strength of weak learnability. *Machine Learning*, 5(2):197–227, 1990.

[11] Robert E. Schapire and Yoram Singer. Improved boosting algorithms using confidence-rated predictions. COLT'98.

[12] V. N. Vapnik. *Estimation of Dependences Based on Empirical Data*. Springer-Verlag, 1982.

[13] Vladimir N. Vapnik. *The Nature of Statistical Learning Theory*. Springer, 1995.